# Bayesian Ensemble Learning

**Hugh A. Chipman**
Department of Mathematics and Statistics
Acadia University
Wolfville, NS, Canada

**Edward I. George**
Department of Statistics
The Wharton School
University of Pennsylvania
Philadelphia, PA 19104-6302

**Robert E. McCulloch**
Graduate School of Business
University of Chicago
Chicago, IL, 60637

## Abstract

We develop a Bayesian "sum-of-trees" model, named BART, where each tree is constrained by a prior to be a weak learner. Fitting and inference are accomplished via an iterative backfitting MCMC algorithm. This model is motivated by ensemble methods in general, and boosting algorithms in particular. Like boosting, each weak learner (i.e., each weak tree) contributes a small amount to the overall model. However, our procedure is defined by a statistical model: a prior and a likelihood, while boosting is defined by an algorithm. This model-based approach enables a full and accurate assessment of uncertainty in model predictions, while remaining highly competitive in terms of predictive accuracy.

## 1 Introduction

We consider the fundamental problem of making inference about an unknown function $f$ that predicts an output $Y$ using a $p$ dimensional vector of inputs $x$ when $Y = f(x) + \epsilon$, $\epsilon \sim N(0, \sigma^2)$. To do this, we consider modelling or at least approximating $f(x) = E(Y \mid x)$, the mean of $Y$ given $x$, by a sum of $m$ regression trees: $f(x) \approx g_1(x) + g_2(x) + \ldots + g_m(x)$ where each $g_i$ denotes a binary regression tree.

The sum-of-trees model is fundamentally an additive model with multivariate components. It is vastly more flexible than a single tree model which does not easily incorporate additive effects. Because multivariate components can easily account for high order interaction effects, a sum-of-trees model is also much more flexible than typical additive models that use low dimensional smoothers as components.

Our approach is fully model based and Bayesian. We specify a prior, and then obtain a sequence of draws from the posterior using Markov chain Monte Carlo (MCMC). The prior plays two essential roles. First, with $m$ chosen large, it restrains the fit of each individual $g_i$ so that the overall fit is made up of many small contributions in the spirit of boosting (Freund & Schapire (1997), Friedman (2001)). Each $g_i$ is a "weak learner". Second, it "regularizes" the model by restraining the overall fit to achieve good bias-variance tradeoff. The prior specification is kept simple and a default choice is shown to have good out of sample predictive performance.

Inferential uncertainty is naturally quantified in the usual Bayesian way: variation in the MCMC draws of $f = \sum g_i$ (evaluated at a set of $x$ of interest) and $\sigma$ indicates our beliefs about plausible values given the data. Note that the depth of each tree is not fixed so that we infer the level of interaction. Our point estimate of $f$ is the average of the draws. Thus, our procedure captures ensemble

learning (in which many trees are combined) both in the fundamental sum-of-trees specification and in the model-averaging used to obtain the estimate.

## 2 The Model

The model consists of two parts: a sum-of-trees model, which we have named BART (Bayesian Additive Regression Trees), and a regularization prior.

### 2.1 A Sum-of-Trees Model

To elaborate the form of a sum-of-trees model, we begin by establishing notation for a single tree model. Let $T$ denote a binary tree consisting of a set of interior node decision rules and a set of terminal nodes, and let $M = \{\mu_1, \mu_2, \ldots, \mu_B\}$ denote a set of parameter values associated with each of the $B$ terminal nodes of $T$. Prediction for a particular value of input vector $x$ is accomplished as follows: If $x$ is associated with terminal node $b$ of $T$ by the sequence of decision rules from top to bottom, it is then assigned the $\mu_b$ value associated with this terminal node. We use $g(x; T, M)$ to denote the function corresponding to $(T, M)$ which assigns a $\mu_b \in M$ to $x$.

Using this notation, and letting $g_i(x) = g(x; T_i, M_i)$, our sum-of-trees model can more explicitly be expressed as

$$Y = g(x; T_1, M_1) + g(x; T_2, M_2) + \cdots + g(x; T_m, M_m) + \epsilon, \tag{1}$$

$$\epsilon \sim N(0, \sigma^2). \tag{2}$$

Unlike the single tree model, when $m > 1$ the terminal node parameter $\mu_i$ given by $g(x; T_j, M_j)$ is merely part of the conditional mean of $Y$ given $x$. Such terminal node parameters will represent interaction effects when their assignment depends on more than one component of $x$ (i.e., more than one variable). Because (1) may be based on trees of varying sizes, the sum-of-trees model can incorporate both direct effects and interaction effects of varying orders. In the special case where every terminal node assignment depends on just a single component of $x$, the sum-of-trees model reduces to a simple additive function.

With a large number of trees, a sum-of-trees model gains increased representation flexibility, which, when coupled with our regularization prior, gives excellent out of sample predictive performance. Indeed, in the examples in Section 4, we set $m$ as large as 200. Note that with $m$ large there are hundreds of parameters of which only $\sigma$ is identified. This is not a problem for our Bayesian analysis. Indeed, this lack of identification is the reason our MCMC mixes well. Even when $m$ is much larger than needed to capture $f$ (effectively, we have an "overcomplete basis") the procedure still works well.

### 2.2 A Regularization Prior

The complexity of the prior specification is vastly simplified by letting the $T_i$ be i.i.d, the $\mu_{i,b}$ (node $b$ of tree $i$) be i.i.d given the set of $T$, and $\sigma$ be independent of all $T$ and $\mu$. Given these independence assumptions we need only choose priors for a single tree $T$, a single $\mu$, and $\sigma$. Motivated by our desire to make each $g(x; T_i, M_i)$ a small contribution to the overall fit, we put prior weight on small trees and small $\mu_{i,b}$.

For the tree prior, we use the same specification as in Chipman, George & McCulloch (1998). In this prior, the probability that a node is nonterminal is $\alpha(1 + d)^{-\beta}$ where $d$ is the depth of the node. In all examples we use the same prior corresponding to the choice $\alpha = .95$ and $\beta = 2$. With this choice, trees with 1, 2, 3, 4, and $\geq 5$ terminal nodes receive prior probability of 0.05, 0.55, 0.28, 0.09, and 0.03, respectively. Note that even with this prior, trees with many terminal nodes can be grown if the data demands it. At any non-terminal node, the prior on the associated decision rule puts equal probability on each available variable and then equal probability on each available rule given the variable.

For the prior on a $\mu$, we start by simply shifting and rescaling $Y$ so that we believe the prior probability that $E(Y \mid x) \in (-.5, .5)$ is very high. We let $\mu \sim N(0, \sigma_\mu^2)$. Given the $T_i$ and an $x$, $E(Y \mid x)$ is the sum of $m$ independent $\mu$'s. The standard deviation of the sum is $\sqrt{m}\, \sigma_\mu$. We choose $\sigma_\mu$ so

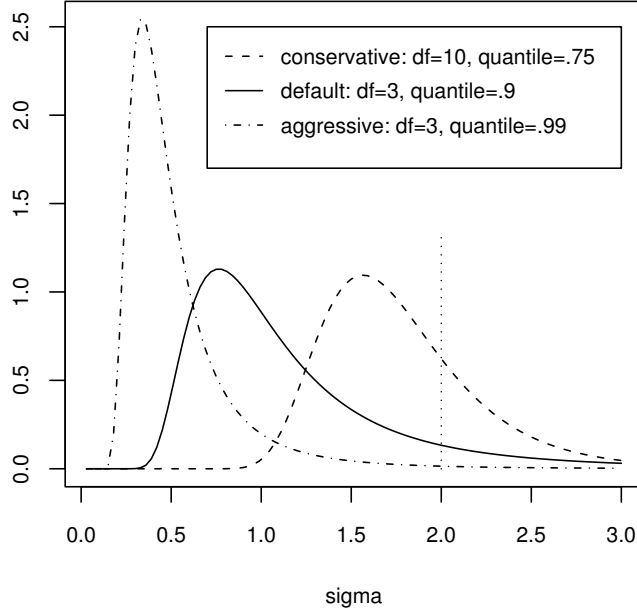

Figure 1: Three priors on $\sigma$ when $\hat{\sigma} = 2$.

that .5 is within $k$ standard deviations of zero: $k\sqrt{m}\sigma_\mu = .5$. For example if $k = 2$ there is a 95% (conditional) prior probability that the mean of $Y$ is in $(-.5, .5)$. $k = 2$ is our default choice and in practice we typically rescale the response $y$ so that its observed values range from -5. to .5. Note that this prior increases the shrinkage of $\mu_{i,b}$ (toward zero) as $m$ increases.

For the prior on $\sigma$ we start from the usual inverted-chi-squared prior: $\sigma^2 \sim \nu\,\lambda/\chi_\nu^2$. To choose the hyperparameters $\nu$ and $\lambda$, we begin by obtaining a "rough overestimate" $\hat{\sigma}$ of $\sigma$. We then pick a degrees of freedom value $\nu$ between 3 and 10. Finally, we pick a value of $q$ such as 0.75, 0.90 or 0.99, and set $\lambda$ so that the $q$th quantile of the prior on $\sigma$ is located at $\hat{\sigma}$, that is $P(\sigma < \hat{\sigma}) = q$. Figure 1 illustrates priors corresponding to three $(\nu, q)$ settings when the rough overestimate is $\hat{\sigma} = 2$. We refer to these three settings, $(\nu, q) = (10, 0.75), (3, 0.90), (3, 0.99)$, as conservative, default and aggressive, respectively. For automatic use, we recommend the default setting $(\nu, q) = (3, 0.90)$ which tends to avoid extremes. Simple data-driven choices of $\hat{\sigma}$ we have used in practice are the estimate from a linear regression or the sample standard deviation of $Y$. Note that this prior choice can be influential. Strong prior beliefs that $\sigma$ is very small could lead to over-fitting.

## 3 A Backfitting MCMC Algorithm

Given the observed data $y$, our Bayesian setup induces a posterior distribution $p((T_1, M_1), \ldots, (T_m, M_m), \sigma \,|\, y)$ on all the unknowns that determine a sum-of-trees model. Although the sheer size of this parameter space precludes exhaustive calculation, the following backfitting MCMC algorithm can be used to sample from this posterior.

At a general level, our algorithm is a Gibbs sampler. For notational convenience, let $T_{(i)}$ be the set of all trees in the sum *except* $T_i$, and similarly define $M_{(i)}$. The Gibbs sampler here entails $m$ successive draws of $(T_i, M_i)$ conditionally on $(T_{(i)}, M_{(i)}, \sigma)$:

$$(T_1, M_1) | T_{(1)}, M_{(1)}, \sigma, y$$
$$(T_2, M_2) | T_{(2)}, M_{(2)}, \sigma, y \tag{3}$$
$$\vdots$$
$$(T_m, M_m) | T_{(m)}, M_{(m)}, \sigma, y,$$

followed by a draw of $\sigma$ from the full conditional:

$$\sigma | T_1, \ldots T_m, M_1, \ldots, M_m, y. \tag{4}$$

Hastie & Tibshirani (2000) considered a similar application of the Gibbs sampler for posterior sampling for additive and generalized additive models with $\sigma$ fixed, and showed how it was a stochastic generalization of the backfitting algorithm for such models. For this reason, we refer to our algorithm as backfitting MCMC. In contrast with the stagewise nature of most boosting algorithms (Freund & Schapire (1997), Friedman (2001), Meek, Thiesson & Heckerman (2002)), the backfitting MCMC algorithm repeatedly resamples the parameters of each learner in the ensemble.

The idea is that given $(T_{(i)}, M_{(i)})$ and $\sigma$ we may subtract the fit from $(T_{(i)}, M_{(i)})$ from both sides of (1) leaving us with a single tree model with known error variance. This draw may be made following the approach of Chipman et al. (1998) or the refinement of Wu, Tjelmeland & West (2007). These methods draw $(T_i, M_i) \mid T_{(i)}, M_{(i)}, \sigma, y$ as $T_i \mid T_{(i)}, M_{(i)}, \sigma, y$ followed by $M_i \mid T_i, T_{(i)}, M_{(i)}, \sigma, y$. The first draw is done by the Metropolis-Hastings algorithm after integrating out $M_i$ and the second is a set of normal draws. The draw of $\sigma$ is easily accomplished by subtracting all the fit from both sides of (1) so the the $\epsilon$ are considered to be observed. The draw is then a standard inverted-chi-squared.

The Metropolis-Hastings draw of $T_i \mid T_{(i)}, M_{(i)}, \sigma, y$ is complex and lies at the heart of our method. The algorithm of Chipman et al. (1998) proposes a new tree based on the current tree using one of four moves. The moves and their associated proposal probabilities are: growing a terminal node (0.25), pruning a pair of terminal nodes (0.25), changing a non-terminal rule (0.40), and swapping a rule between parent and child (0.10). Although the grow and prune moves change the implicit dimensionality of the proposed tree in terms of the number of terminal nodes, by integrating out $M_i$ from the posterior, we avoid the complexities associated with reversible jumps between continuous spaces of varying dimensions (Green 1995).

We initialize the chain with $m$ single node trees, and then iterations are repeated until satisfactory convergence is obtained. At each iteration, each tree may increase or decrease the number of terminal nodes by one, or change one or two decision rules. Each $\mu$ will change (or cease to exist or be born), and $\sigma$ will change. It is not uncommon for a tree to grow large and then subsequently collapse back down to a single node as the algorithm iterates. The sum-of-trees model, with its abundance of unidentified parameters, allows for "fit" to be freely reallocated from one tree to another. Because each move makes only small incremental changes to the fit, we can imagine the algorithm as analogous to sculpting a complex figure by adding and subtracting small dabs of clay.

Compared to the single tree model MCMC approach of Chipman et al. (1998), our backfitting MCMC algorithm mixes dramatically better. When only single tree models are considered, the MCMC algorithm tends to quickly gravitate toward a single large tree and then gets stuck in a local neighborhood of that tree. In sharp contrast, we have found that restarts of the backfitting MCMC algorithm give remarkably similar results even in difficult problems. Consequently, we run one long chain rather than multiple starts.

In some ways backfitting MCMC is a stochastic alternative to boosting algorithms for fitting linear combinations of trees. It is distinguished by the ability to sample from a posterior distribution. At each iteration, we get a new draw

$$f^* = g(x; T_1, M_1) + g(x; T_2, M_2) + \ldots + g(x; T_m, M_m) \tag{5}$$

corresponding to the draw of $T_j$ and $M_j$. These draws are a (dependent) sample from the posterior distribution on the "true" $f$. Rather than pick the "best" $f^*$ from these draws, the set of multiple draws can be used to further enhance inference. We estimate $f$ by the posterior mean of $f$ which is approximated by averaging the $f^*$ over the draws. Further, we can gauge our uncertainty about the actual underlying $f$ by the variation across the draws. For example, we can use the 5% and 95% quantiles of $f^*(x)$ to obtain 90% posterior intervals for $f(x)$.

## 4 Examples

In this section we illustrate the potential of our Bayesian ensemble procedure BART in a large experiment using 42 datasets. The data are a subset of 52 sets considered by Kim, Loh, Shih & Chaudhuri

| Method | Parameter | Values considered |
|---|---|---|
| Lasso | shrinkage (in range 0-1) | 0.1, 0.2, ..., 1.0 |
| Gradient Boosting | # of trees | 50, 100, 200 |
| | Shrinkage (multiplier of each tree added) | 0.01, 0.05, 0.10, 0.25 |
| | Max depth permitted for each tree | 1, 2, 3, 4 |
| Neural Nets | # hidden units | see text |
| | Weight decay | .0001,.001, .01, .1, 1, 2, 3 |
| Random Forests | # of trees | 500 |
| | % variables sampled to grow each node | 10, 25, 50, 100 |
| *BART-cv* | Sigma prior: $(\nu, q)$ combinations | (3,0.90), (3,0.99), (10,0.75) |
| | # trees | 50, 200 |
| | $\mu$ Prior: $k$ value for $\sigma_\mu$ | 2, 3, 5 |

Table 1: Operational parameters for the various competing models.

(2007). Ten datasets were excluded either because Random Forests was unable to use over 32 categorical predictors, or because a single train/test split was used in the original paper. All datasets correspond to regression problems with between 3 and 28 numeric predictors and 0 to 6 categorical predictors. Categorical predictors were converted into 0/1 indicator variables corresponding to each level. Sample sizes vary from 96 to 6806 observations.

As competitors we considered linear regression with L1 regularization (the Lasso) (Efron, Hastie, Johnstone & Tibshirani 2004) and four black-box models: Friedman's (2001) gradient boosting, random forests (Breiman 2001), and neural networks with one layer of hidden units. Implementation details are given in Chipman, George & McCulloch (2006). Tree models were not considered, since they tend to sacrifice predictive performance for interpretability.

We considered two versions of our Bayesian ensemble procedure BART. In *BART-cv*, the prior hyperparameters $(\nu, q, k, m)$ were treated as operational parameters to be tuned via cross-validation. In *BART-default*, we set $(\nu, q, k, m) = (3, 0.90, 2, 200)$. For both *BART-cv* and *BART-default*, all specifications of the quantile $q$ were made relative to the least squares linear regression estimate $\hat{\sigma}$, and the number of burn-in steps and MCMC iterations used were determined by inspection of a single long run. Typically 200 burn-in steps and 1000 iterations were used.

With the exception of *BART-default* (which has no tuning parameters), all free parameters in learners were chosen via 5-fold cross-validation within the training set. The parameters considered and potential levels are given in Table 1. The levels used were chosen with a sufficientlly wide range that the optimal value was not at an extreme of the candidate values in most problems. Neural networks are the only model whose operational parameters need additional explanation. In that case, the number of hidden units was chosen in terms of the implied number of weights, rather than the number of units. This design choice was made because of the widely varying number of predictors across problems, which directly impacts the number of weights. A number of hidden units was chosen so that there was a total of roughly $u$ weights, with $u = 50, 100, 200, 500$ or $800$. In all cases, the number of hidden units was further constrained to fall between 3 and 30. For example, with 20 predictors we used 3, 8 and 21 as candidate values for the number of hidden units.

The models were compared with 20 replications of the following experiment. For each replication, we randomly chose 5/6 of the data as a training set and the remaining 1/6 was used for testing. As mentioned above, 5-fold cv was used within each training set. In each of the 42 datasets, the response was minimally preprocessed, applying a log or square root transformation if this made the histogram of observed responses more bell-shaped. In about half the cases, a log transform was used to reduce a right tail. In one case (Fishery) a square root transform was most appropriate. Finally, in order to enable performance comparisons across all datasets, after possible nonlinear transformation, the resultant response was scaled to have sample mean 0 and standard deviation 1 prior to any train/test splitting.

A total of $42 \times 20 = 840$ experiments were carried out. Results across these experiments are summarized in Table 2, which gives mean RMSE values and Figure 2, which summarizes relative performance using boxplots. In Figure 2, the relative performances are calculated as follows: In

| Method | *BART-cv* | Boosting | *BART-default* | Random Forest | Neural Net | Lasso |
|---|---|---|---|---|---|---|
| RMSE | 0.5042 | 0.5089 | 0.5093 | 0.5097 | 0.5160 | 0.5896 |

Table 2: Average test set RMSE values for each learner, combined across 20 train/test replicates of 42 datasets. The only statistically significant difference is Lasso versus the other methods.

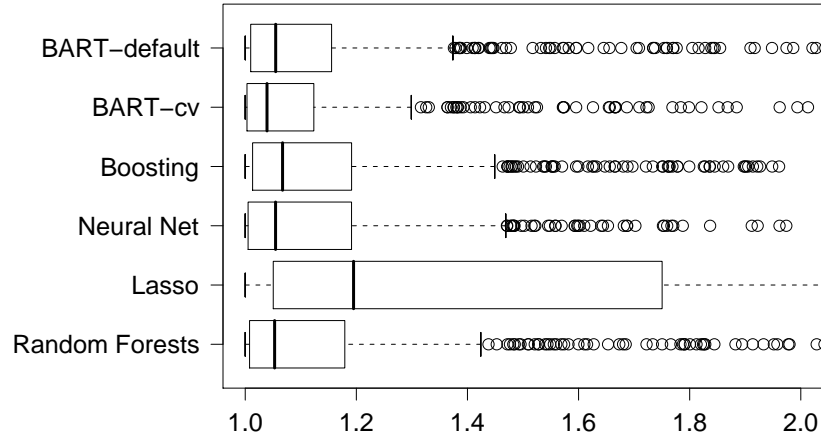

Figure 2: Test set RMSE performance relative to best (ratio of 1 means minimum RMSE test error). Results are across 20 replicates in each of 42 datasets. Boxes indicate middle 50% of runs. Each learner has the following percentage of ratios larger than 2.0, which are not plotted above: Neural net: 5%, *BART-cv*: 6%, *BART-default* and Boosting: 7%, Random forests 10% and Lasso 21%.

each of the 840 experiments, the learner with smallest RMSE was identified. The relative ratio for each learner is the raw RMSE divided by the smallest RMSE. Thus a relative RMSE of 1 means that the learner had the best performance in a particular experiment. The central box gives the middle 50% of the data, with the median indicated by a vertical line. The "whiskers" of the plot extend to 1.5 times the box width, or the range of values, whichever comes first. Extremes outside the whiskers are given by individual points. As noted in the caption, relative RMSE ratios larger than 2.0 are not plotted.

BART has the best performance, although all methods except the Lasso are not significantly different. The strong performance of our "default" ensemble is especially noteworthy, since it requires no selection of operational parameters. That is, cross-validation is not necessary. This results in a huge computational savings, since under cross-validation, the number of times a learner must be trained is equal to the number of settings times the number of folds. This can easily be 50 (e.g. 5 folds by 10 settings), and in this experiment it was 90!

*BART-default* is in some sense the "clear winner" in this experiment. Although average predictive performance was indistinguishable from the other models, it does not require cross-validation. Moreover, the use of cross-validation makes it impossible to interpret the MCMC output as valid uncertainty bounds. Not only is the default version of BART faster, but it also provides valid statistical inference, a benefit not available to any of the other learners considered.

To further stress the benefit of uncertainty intervals, we report some more detailed results in the analysis of one of the 42 datasets, the Boston Housing data. We applied BART to all 506 observations of the Boston Housing data using the default setting $(\nu, q, k, m) = (3, 0.90, 2, 200)$ and the linear regression estimate $\hat{\sigma}$ to anchor $q$. At each of the 506 predictor values $x$, we used 5% and 95% quantiles of the MCMC draws to obtain 90% posterior intervals for $f(x)$. An appealing feature of these posterior intervals is that they widen when there is less information about $f(x)$. To roughly illustrate this, we calculated Cook's distance diagnostic $D_x$ for each $x$ (Cook 1977) based on a linear least squares regression of $y$ on $x$. Larger $D_x$ indicate more uncertainty about predicting $y$ with a

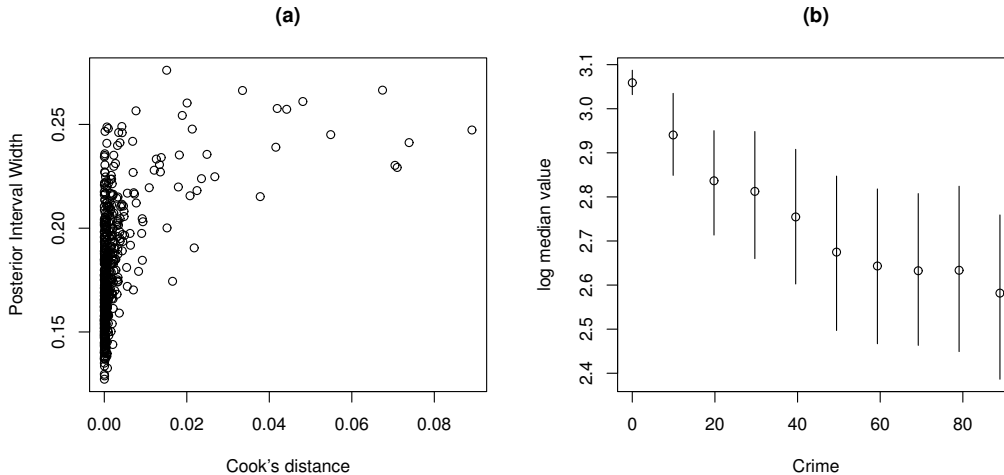

Figure 3: Plots from a single run of the Bayesian Ensemble model on the full Boston dataset. (a) Comparison of uncertainty bound widths with Cook's distance measure. (b) Partial dependence plot for the effect of `crime` on the response (log median property value), with 90% uncertainty bounds.

linear regression at $x$. To see how the width of the 90% posterior intervals corresponded to $D_x$, we plotted them together in Figure 3(a). Although the linear model may not be strictly appropriate, the plot is suggestive: all points with large $D_x$ values have wider uncertainty bounds.

Uncertainty bounds can also be used in graphical summaries such as a partial dependence plot (Friedman 2001), which shows the effect of one (or more) predictor on the response, margining out the effect of other predictors. Since BART provides posterior draws for $f(x)$, calculation of a posterior distribution for the partial dependence function is straightforward. Computational details are provided in Chipman et al. (2006). For the Boston Housing data, Figure 3(b) shows the partial dependence plot for `crime`, with 90% posterior intervals. The vast majority of data values occur for `crime` $< 5$, causing the intervals to widen as `crime` increases and the data become more sparse.

## 5 Discussion

Our approach is a fully Bayesian approach to learning with ensembles of tree models. Because of the nature of the underlying tree model, we are able to specify simple, effective priors and fully exploit the benefits of Bayesian methodology. Our prior provides the regularization needed to obtain good predictive performance. In particular, our default prior, which is minimially dependent on the data, performs well compared to other methods which rely on cross-validation to pick model parameters. We obtain inference in the natural Bayesian way from the variation in the posterior draws. While predictive performance in always our first goal, many researchers want to interpret the results. In this case, gauging the inferential uncertainty is essential. No other competitive methods do this in a convenient way.

Chipman et al. (2006) and Abreveya & McCulloch (2006) provide further evidence of the predictive performance of our approach. In addition Abreveya & McCulloch (2006) illustrate the ability of our method to uncover interesting interaction effects in a real example. Chipman et al. (2006) and and Hill & McCulloch (2006) illustrate the inferential capabilities. Posterior intervals are shown to have good frequentist coverage. Chipman et al. (2006) also illustrates the method's ability to obtain inference in the very difficult "big p, small n" problem, where there are few observations and many potential predictors.

A common concern with Bayesian approaches is sensitivity to prior parameters. Chipman et al. (2006) found that results were robust to a reasonably wide range of prior parameters, including $\nu, q, \sigma_\mu$, as well as the number of trees, $m$. $m$ needs to be large enough to provide enough complexity

to capture $f$, but making $m$ "too large" does not appreciably degrade accuracy (although it does make it slower to run). Chipman et al. (2006) provide guidelines for choosing $m$.

In practice, the stability of the MCMC makes the method easy to use. Typcially, it burns-in rapidly. If the method is run twice with different seeds the same results are obtained both for fit and inference.

Code is publicly available in the R-package BayesTree.

**Acknowledgments**

The authors would like to thank three anonymous referees, whose comments improved an earlier draft, and Wei-Yin Loh who generously provided the datasets used in the experiment. This research was supported by the Natural Sciences and Engineering Research Council of Canada, the Canada Research Chairs program, the Acadia Centre for Mathematical Modelling and Computation, the University of Chicago Graduate School of Business, NSF grant DMS 0605102 and by NIH/NIAID award AI056983.

# References

Abreveya, J. & McCulloch, R. (2006), Reversal of fortune: a statistical analysis of penalty calls in the national hockey league, Technical report, Purdue University.

Breiman, L. (2001), 'Random forests', *Machine Learning* **45**, 5–32.

Chipman, H. A., George, E. I. & McCulloch, R. E. (1998), 'Bayesian CART model search (C/R: p948-960)', *Journal of the American Statistical Association* **93**, 935–948.

Chipman, H. A., George, E. I. & McCulloch, R. E. (2006), BART: Bayesian additive regression trees, Technical report, University of Chicago.

Cook, R. D. (1977), 'Detection of influential observations in linear regression', *Technometrics* **19**(1), 15–18.

Efron, B., Hastie, T., Johnstone, I. & Tibshirani, R. (2004), 'Least angle regression', *Annals of Statistics* **32**, 407–499.

Freund, Y. & Schapire, R. E. (1997), 'A decision-theoretic generalization of on-line learning and an application to boosting', *Journal of Computer and System Sciences* **55**, 119–139.

Friedman, J. H. (2001), 'Greedy function approximation: A gradient boosting machine', *The Annals of Statistics* **29**, 1189–1232.

Green, P. J. (1995), 'Reversible jump MCMC computation and Bayesian model determination', *Biometrika* **82**, 711–732.

Hastie, T. & Tibshirani, R. (2000), 'Bayesian backfitting (with comments and a rejoinder by the authors', *Statistical Science* **15**(3), 196–223.

Hill, J. L. & McCulloch, R. E. (2006), Bayesian nonparametric modeling for causal inference, Technical report, Columbia University.

Kim, H., Loh, W.-Y., Shih, Y.-S. & Chaudhuri, P. (2007), 'Visualizable and interpretable regression models with good prediction power', *IEEE Transactions: Special Issue on Data Mining and Web Mining*. In press.

Meek, C., Thiesson, B. & Heckerman, D. (2002), Staged mixture modelling and boosting, Technical Report MS-TR-2002-45, Microsoft Research.

Wu, Y., Tjelmeland, H. & West, M. (2007), 'Bayesian CART: Prior specification and posterior simulation', *Journal of Computational and Graphical Statistics*. In press.
